# Modeling Neuronal Interactivity using Dynamic Bayesian Networks

**Lei Zhang**†,‡, **Dimitris Samaras**†, **Nelly Alia-Klein**‡, **Nora Volkow**‡, **Rita Goldstein**‡
† Computer Science Department, SUNY at Stony Brook, Stony Brook, NY
‡ Medical Department, Brookhaven National Laboratory, Upton, NY

## Abstract

Functional Magnetic Resonance Imaging (fMRI) has enabled scientists to look into the active brain. However, interactivity between functional brain regions, is still little studied. In this paper, we contribute a novel framework for modeling the interactions between multiple active brain regions, using Dynamic Bayesian Networks (DBNs) as generative models for brain activation patterns. This framework is applied to modeling of neuronal circuits associated with reward. The novelty of our framework from a Machine Learning perspective lies in the use of DBNs to reveal the brain connectivity and interactivity. Such interactivity models which are derived from fMRI data are then validated through a group classification task. We employ and compare four different types of DBNs: Parallel Hidden Markov Models, Coupled Hidden Markov Models, Fully-linked Hidden Markov Models and Dynamically Multi-Linked HMMs (DML-HMM). Moreover, we propose and compare two schemes of learning DML-HMMs. Experimental results show that by using DBNs, group classification can be performed even if the DBNs are constructed from as few as 5 brain regions. We also demonstrate that, by using the proposed learning algorithms, different DBN structures characterize drug addicted subjects vs. control subjects. This finding provides an independent test for the effect of psychopathology on brain function. In general, we demonstrate that incorporation of computer science principles into functional neuroimaging clinical studies provides a novel approach for probing human brain function.

## 1. Introduction

Functional Magnetic Resonance Imaging (fMRI) has enabled scientists to look into the active human brain [1] by providing sequences of 3D brain images with intensities representing blood oxygenation level dependent (BOLD) regional activations. This has revealed exciting insights into the spatial and temporal changes underlying a broad range of brain functions, such as how we see, feel, move, understand each other and lay down memories. This fMRI technology offers further promise by imaging the dynamic aspects of the functioning human brain. Indeed, fMRI has encouraged a growing interest in revealing brain connectivity and interactivity within the neuroscience community. It is for example understood that a dynamically managed goal directed behavior requires neural control mechanisms orchestrated to select the appropriate and task-relevant responses while inhibiting irrelevant or inappropriate processes [12]. To date, the analyses and interpretation of fMRI data that are most commonly employed by neuroscientists depend on the

cognitive-behavioral probes that are developed to tap regional brain function. Thus, brain responses are a-priori labeled based on the putative underlying task condition and are then used to separate a priori defined groups of subjects. In recent computer science research [18][13][3][19], machine learning methods have been applied for fMRI data analysis. However, in these approaches information on the connectivity and interactivity between brain voxels is discarded and brain voxels are assumed to be independent, which is an inaccurate assumption (see use of statistical maps [3][19] or the mean of each fMRI time interval[13]). In this paper, we exploit Dynamic Bayesian Networks for modeling dynamic (i.e., connecting and interacting) neuronal circuits from fMRI sequences. We suggest that through incorporation of graphical models into functional neuroimaging studies we will be able to identify neuronal patterns of connectivity and interactivity that will provide invaluable insights into basic emotional and cognitive neuroscience constructs. We further propose that this interscientific incorporation may provide a valid tool where objective brain imaging data are used for the clinical purpose of diagnosis of psychopathology. Specifically, in our case study we will model neuronal circuits associated with reward processing in drug addiction. We have previously shown loss of sensitivity to the relative value of money in cocaine users [9]. It has also been previously highlighted that the complex mechanism of drug addiction requires the connectivity and interactivity between regions comprising the mesocorticolimbic circuit [12][8]. However, although advancements have been made in studying this circuit's role in inhibitory control and reward processing, inference about the connectivity and interactivity of these regions is at best indirect. Dynamical causal models have been compared in [16]. Compared with dynamic causal models, DBNs admit a class of nonlinear continuous-time interactions among the hidden states and model both causal relationships between brain regions and temporal correlations among multiple processes, useful for both classification and prediction purposes.

Probabilistic graphical models [14][11] are graphs in which nodes represent random variables, and the (lack of) arcs represent conditional independence assumptions. In our case, interconnected brain regions can be considered as nodes of a probabilistic graphical model and interactivity relationships between regions are modeled by probability values on the arcs (or the lack of) between these nodes. However, the major challenge in such a machine learning approach is the choice of a particular structure that models connectivity and interactivity between brain regions in an *accurate* and *efficient* manner. In this work, we contribute a framework of exploiting Dynamic Bayesian Networks to model such a structure for the fMRI data. More specifically, instead of modeling each brain region in isolation, we aim to model the interactive pattern of multiple brain regions. Furthermore, the revealed functional information is validated through a group classification case study: separating drug addicted subjects from healthy non-drug-using controls based on trained Dynamic Bayesian Networks. Both conventional BBNs and HMMs are unsuitable for modeling activities underpinned not only by causal but also by clear temporal correlations among multiple processes [10], and Dynamic Bayesian Networks [5][7] are required. Since the state of each brain region is not known (only observations of activation exist), it can be thought of as a hidden variable[15]. An intuitive way to construct a DBN is to extend a standard HMM to a set of interconnected multiple HMMs. For example, Vogler et al. [17] proposed Parallel Hidden Markov Models (PaHMMs) that factorize state space into multiple independent temporal processes without causal connections in-between. Brand et al. [2] exploited Coupled Hidden Markov Models (CHMMs) for complex action recognitions. Gong et al. [10] developed a Dynamically Multi-Linked Hidden Markov Model (DML-HMM) for the recognition of group activities involving multiple different object events in a noisy outdoor scene. This model is the only one of those models that learns both the structure and parameters of the graphical model, instead of presuming a structure (possibly inaccurate) given the lack of knowledge of human brain connectivity. In order to model the dynamic neuronal circuits underlying reward processing in the human brains, we explore and compare the above DBNs. We propose and compare two learning schemes of

DML-HMMs, one is greedy structure search (Hill-Climbing) and the other is Structural Expectation-Maximization (SEM).

To our knowledge, this is the first time that Dynamic Bayesian Networks are exploited in modeling the connectivity and interactivity among brain regions activated during a fMRI study. Our current experimental classification results show that by using DBNs, group classification can be performed even if the DBNs are constructed from as few as 5 brain regions. We also demonstrate that, by using the proposed learning algorithms, different DBN structures characterize drug addicted subjects vs. control subjects which provides an independent test for the effects of psychopathology on brain function. From the machine learning point of view, this paper provides an innovative application of Dynamic Bayesian Networks in modeling dynamic neuronal circuits. Furthermore, since the structures to be explored are exclusively represented by hidden (cannot be observed directly) states and their interconnecting arcs, the structure learning of DML-HMMs poses a greater challenge than other DBNs [5]. From the neuroscientific point of view, drug addiction is a complex disorder characterized by compromised inhibitory control and reward processing. However, individuals with compromised mechanisms of control and reward are difficult to identify unless they are directly subjected to challenging conditions. Modeling the interactive brain patterns is therefore essential since such patterns may be unique to a certain psychopathology and could hence be used for improving diagnosis and prevention efforts (e.g., diagnosis of drug addiction, prevention of relapse or craving). In addition, the development of this framework can be applied to further our understanding of other human disorders and states such as those impacting insight and awareness, that similarly to drug addiction are currently identified based mostly on subjective criteria and self-report.

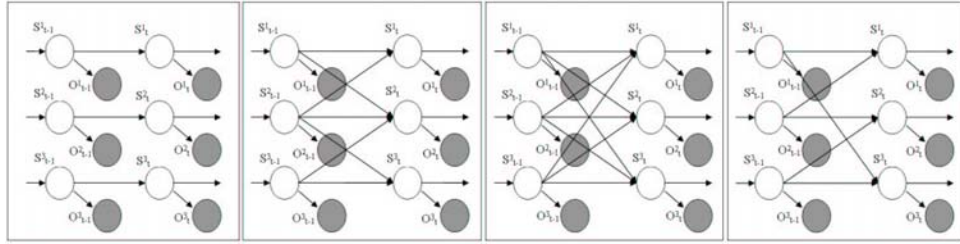

Figure 1: Four types of Dynamic Bayesian Networks: PaHMM, CHMM, FHMM and DML-HMM.

## 2. Dynamic Bayesian Networks

In this section, we will briefly describe the general framework of Dynamic Bayesian Networks. DBNs are Bayesian Belief Networks that have been extended to model the stochastic evolution of a set of random variables over time [5][7]. As described in [10], a DBN $B$ can be represented by two sets of parameters $(m, \Theta)$ where the first set $m$ represents the structure of the DBN including the number of hidden state variables $S$ and observation variables $O$ per time instance, the number of states for each hidden state variable and the topology of the network (set of directed arcs connecting the nodes). More specifically, the $i$th hidden state variable and the $j$th observation variable at time instance $t$ are denoted as $S_t^{(i)}$ and $O_t^{(j)}$ with $i \in \{1, ..., N_h\}$ and $j \in \{1, ..., N_o\}$, $N_h$ and $N_o$ are the number of hidden state variables and observation variables respectively. The second set of parameters $\Theta$ includes the state transition matrix $A$, the observation matrix $B$ and a matrix $\pi$ modeling the initial state distribution $P(S_1^i)$. More specifically, $A$ and $B$ quantify the transition models $P(S_t^{(i)}|Pa(S_t^{(i)}))$ and observation models $P(O_t^{(i)}|Pa(O_t^{(i)}))$ respectively where $Pa(S_t^{(i)})$ are the parents of $S_t^{(i)}$ (similarly $Pa(O_t^{(i)})$ for observations). In this paper, we will examine four types of DBNs: Parallel Hidden Markov Models (PaHMM) [17], Coupled Hidden Markov Models (CHMM)[2], Fully Connected Hidden Markov Models (FHMM) and

Dynamically Multi-Linked Hidden Markov Models (DML-HMM)[10] as shown in Fig 1 where observation nodes are shown as shaded circles, hidden nodes as clear circles and the causal relationships among hidden state variables are represented by the arcs between hidden nodes. Notice that the first three DBNs are essentially three special cases of the DML-HMM.

## 2.1. Learning of DBNs

Given the form of DBNs in the previous sections, there are two learning problems that must be solved for real-world applications: 1) **Parameter Learning:** assuming fixed structure, given the training sequences of observations $O$, how we adjust the model parameters $B = (m, \Theta)$ to maximize $P(O|B)$; 2) **Structure Learning:** for DBNs with unknown structure (i.e. DML-HMMs), how we learn the structure from the observation $O$. Parameter learning has been well studied in [17][2]. Given fixed structure, parameters can be learned iteratively using Expectation-Maximization (EM). The E step, which involves the inference of hidden states given parameters, can be implemented using an exact inference algorithm such as the junction tree algorithm. Then the parameters and maximal likelihood $L(\Theta)$ can be computed iteratively from the M step.

In [10], the DML-HMM was selected from a set of candidate structures, however the selection of candidate structure is non-trivial for most applications including brain region connectivity. For a DML-HMM with $N$ hidden nodes, the total number of different structures is $2^{N^2-N}$, thus it is impossible to conduct an exhaustive search in most cases. The learning of DBNs involving both parameter learning and structure learning has been discussed in [5], where the scoring rules for standard probabilistic networks were extended to the dynamic case and the Structural EM (SEM) algorithm was developed for structure learning when some of the variables are hidden. The structure learning of DML-HMMs is more challenging since the structures to be explored are exclusively represented by the hidden states and none of them can be directly observed. In the following, we will explain two learning schemes for the DML-HMMs. One standard way is to perform parametric EM within an outer-loop structural search. Thus, our first scheme is to use an outer-loop of the Hill-Climbing algorithm (DML-HMM-HC). For each step of the algorithm, from the current DBN, we first compute a neighbor list by adding, deleting, or reversing one arc. Then we perform parameter learning for each of the neighbors and go to the neighbor with the minimum score until there is no neighbor whose score is higher than the current DBN. Our second learning scheme is similar to the Structural EM algorithm [5] in the sense that the structural and parametric modification are performed within a single EM process. As described in [5][4], a structural search can be performed efficiently given complete observation data. However, as we described above, the structure of DML-HMMs are represented by the hidden states which can not be observed directly. Hence, we develop the DML-HMM-SEM algorithm as follows: given the current structure, we first perform a parameter learning and then, for each training data, we compute the Most Probable Explanation (MPE), which computes the most likely value for each hidden node (similar to Viterbi in standard HMM). The MPE thus provides a complete estimation of the hidden states and a complete-data structural search [4] is then performed to find the best structure. We perform learning iteratively until the structure converges. In this scheme, the structural search is performed in the inner loop thus making the learning more efficient. Pseudo-codes of both learning schemes are described in Table 1. In this paper, we use Schwarz's Bayesian Information Criterion ($BIC$): $BIC = -2 \log L(\Theta_B) + K_B \log N$ as our score function where for a DBN $B$, $L(\Theta_B)$ is the maximal likelihood under $B$, $K_B$ is the dimension of the parameters of $B$ and $N$ is the size of the training data. Theoretically, the DML-HMM-SEM algorithm is not guaranteed to converge since for the same training data, the most probably explanations $(S_i, S_j)$ of two DML-HMMs $B_i, B_j$ might be different. In the worst case, oscillation between two structures is possible. To guarantee halting of the algorithm, a loop detector can be added so that, once any structure is selected in a second

time, we stop the learning and select the structure with the minimum score visited during the searching. However, in our experiments, the learning algorithm always converged in a few steps.

| Procedure **DML-HMM-HC** | Procedure **DML-HMM-SEM** |
|---|---|
| $Initial\_Model(B_0)$; | $Initial\_Model(B_0)$; |
| Loop $i = 0, 1, ...$ until convergence: | Loop $i = 0, 1, ...$ until convergence: |
|   $[B_i', score_i^0] = Learn\_Parameter(B_i)$; |   $[B_i', score_i^0] = Learn\_Parameter(B_i)$; |
|   $B_i^{1...J} = Generate\_Neighbors(B_i)$; |   $S = Most\_Prob\_Expl(B_i', O)$; |
|   for j=1..J |   $B_i^{max} = Find\_Best\_Struct(S)$; |
|     $[B_i^{j'}, score_i^j] = Learn\_Parameter(B_i^j)$; |   if $B_i^{max} == B_i'$ |
|   $j = Find\_Minscore(score_i^{1...J})$; |     return $B_i'$; |
|   if $(score_i^j > score_i^0)$ |   else |
|     return $B_i'$; |     $B_{i+1} = B_i^{max}$; |
|   else | |
|     $B_{i+1} = B_i^j$; | |

Table 1: Two schemes of learning DML-HMMs: the left column lists the DML-HMM-HC scheme and the right column lists the DML-HMM-SEM scheme.

## 3. Modeling Reward Neuronal Circuits: A Case Study

In this section, we will describe our case study of modeling ***Reward Neuronal Circuits***: by using DBNs, we aim to model the interactive pattern of multiple brain regions for the neuropsychological problem of sensitivity to the relative value of money. Furthermore, we will examine the revealed functional information encapsulated in the trained DBNs through a group classification study: separating drug addicted subjects from healthy non-drug-using controls based on trained DBNs.

### 3.1. Data Collection and Preprocess

In our experiments, data were collected to study the neuropsychological problem of loss of sensitivity to the relative value of money in cocaine users[9]. MRI studies were performed on a 4T Varian scanner and all stimuli were presented using LCD-goggles connected to a PC. Human participants pressed a button or refrained from pressing based on a picture shown to them. They received a monetary reward if they performed correctly. Specifically, three runs were repeated twice (T1, T2, T3; and T1R, T2R, T3R) and in each run, there were three monetary conditions (high money, low money, no money) and a baseline condition; the order of monetary conditions was pseudo-randomized and identical for all participants. Participants were informed about the monetary condition by a 3-sec instruction slide, presenting the stimuli: $0.45, $0.01 or $0.00. Feedback for correct responses in each condition consisted of the respective numeral designating the amount of money the subject has earned if correct or the symbol (X) otherwise. To simulate real-life motivational salience, subjects could gain up to $50 depending on their performance on this task. 16 cocaine dependent individuals, 18-55 years of age, in good health, were matched with 12 non-drug-using controls on sex, race, education and general intellectual functioning. Statistical Parametric Mapping (SPM)[6] was used for fMRI data preprocessing (realignment, normalization/registration and smoothing) and statistical analyses.

### 3.2. Feature Selection and Neuronal Circuit Modeling

The fMRI data are extremely high dimensional (i.e. $53 \times 63 \times 46$ voxels per scan). Prior to training the DBN, we selected 5 brain regions: Left Inferior Frontal Gyrus (Left IFG), Prefrontal Cortex (PFC, including lateral and medial dorsolateral PFC and the anterior cingulate), Midbrain (including substantia nigra), Thalamus and Cerebellum. These regions were selected based on prior SPM analyses random-effects analyses (ANOVA) where the goal was to differentiate effect of money (high, low, no) from the effect of group (cocaine,

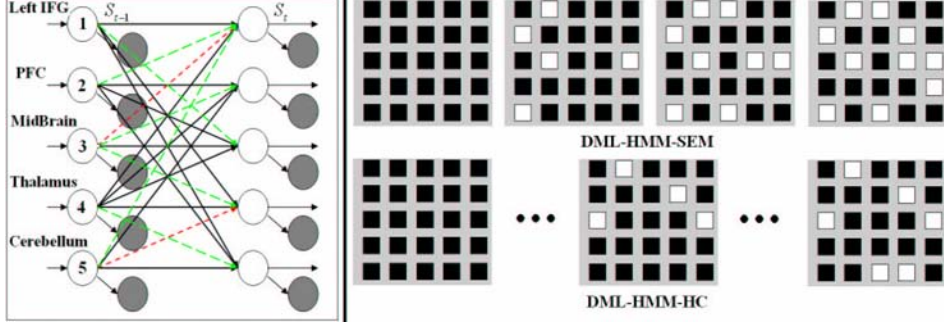

Figure 2: Learning processes and learned structures from two algorithms. The leftmost column demonstrates two (superimposed) learned structures where light gray dashed arcs (long dash) are learned from DML-HMM-HC, dark gray dashed arcs (short dash) from DML-HMM-SEM and black solid arcs from both. The right columns shows the transient structures of the learning processes of two algorithms where black represents existence of arc and white represents no arc.

control) on all regions that were activated to monetary reward in all subjects. In all these five regions, the monetary main effect was significant as evidenced by region of interest follow-up analyses. Of note is the fact that these five regions are part of the mesocorticolimbic reward circuit, previously implicated in addiction. Each of the above brain regions is presented by a $k$-D feature vector where $k$ is the number of brain voxels selected in this brain region (i.e. $k = 3$ for Left IFG and $k = 8$ for PFC). After feature selection, a DML-HMM with 5 hidden nodes can be learned as described in Sec. 2 from the training data. The leftmost image in Fig. 2 shows two superimposed possible structures of such DML-HMMs. The causal relationships discovered among different brain regions are embodied in the topology of the DML-HMM. Each of the five hidden variables has two states (activated or not) and each continuous observation variable (given by a $k$-D feature vector) represents the observed activation of each brain region. The Probabilistic Distribution Function (PDF) of each observation variable is a mixture of Gaussians conditioned by the state of its discrete parent node.

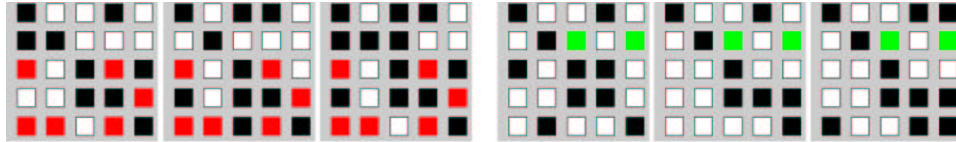

Figure 3: Left three images shows the structures learned from the 3 subsets of Group C and the right three images shows those learned from subsets of Group S. Figure shows that some arcs consistently appeared in Group C but not consistently in Group S (marked in dark gray) and vice versa (marked in light gray), which implies such group differences in the interactive brain patterns may correspond to the loss of sensitivity to the relative value of money in cocaine users.

## 4. Experiments and Results

We collected fMRI data of 16 drug addicted subjects and 12 control subjects, 6 runs per participant. Due to head motion, some data could not be used. In our experiments, we used a total of 152 fMRI sequences (87 scans per sequence) with 86 sequences for the drug addicted subjects (Group S) and 66 for control subjects (Group C).

First we compare the two learning schemes for DML-HMMs proposed in Sec. 2. Fig. 2 demonstrates the learning process (initialized with the FHMM) for drug addicted subjects. The leftmost column shows two learned structures where red arcs are learned from DML-HMM-HC, green arcs from DML-HMM-SEM and black arcs from both. The right columns show the learning processes of DML-HMM-SEM (top) and DML-HMM-HC (bottom) with

black representing existence of arc and white representing no arc. Since in DML-HMM-SEM, structure learning is in the inner loop, the learning process is much faster than that of DML-HMM-HC. We also compared the BIC scores of the learned structures and we found DML-HMM-SEM selected better structures than DML-HMM-HC.

It is also very interesting to examine the structure learning processes by using different training data. For each participant group, we randomly separated the data set into three subsets and trained DBNs are reported in Fig. 3 where the left three images show the structures learned from the 3 subsets of Group C and the right three images show those learned from subsets of Group S. In Fig. 3, we found the learned structures of each group are similar. We also found that some arcs consistently appeared in Group C but not consistently in Group S (marked in red) and vice versa (marked in green), which implies such group differences in the interactive brain patterns may correspond to the loss of sensitivity to the relative value of money in cocaine users. More specifically, in Fig. 3, the average intra-group similarity scores were 80% and 78.3%, while cross-group similarity was 56.7%.

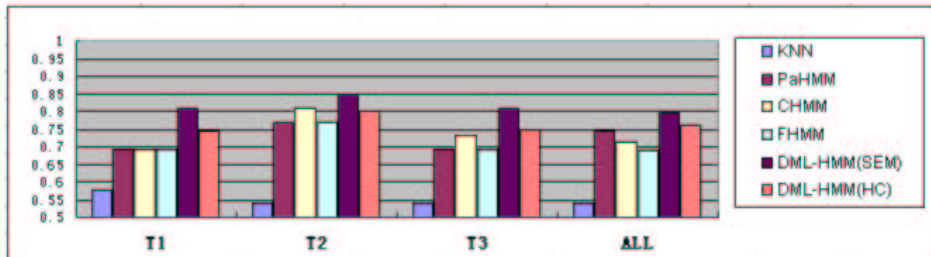

Figure 4: Classification results: All DBN methods significantly improved classification rates compared to K-Nearest Neighbor with DML-HMM performing best.

The second set of experiments was to apply the trained DBNs for group classification. In our data collection, there were 6 runs of fMRI collection: T1, T2, T3, T1R, T2R and T3R with the latter latter repeating the former three, grouped into 4 data sets $\{T1, T2, T3, ALL\}$ with $ALL$ containing all the data. We performed classification experiments on each of the 4 data sets where the data were randomly divided into a training set and a testing set of equal size. During training, the described four DBN type were employed using the training set while during the learning of DML-HMMs, different initial structures (PaHMM, CHMM, FHMM) were used and the structure with the minimum BIC score was selected from the three learned DML-HMMs. For each model, two DBNs $\{B_c, B_s\}$ were trained on the training data of Group C and Group S respectively. During testing, for each testing fMRI sequence $O_{test}$, we computed two likelihoods $P_c^{test} = P(O_{test}|B_c)$ and $P_s^{test} = P(O_{test}|B_s)$ using the two trained DBNs. Since the two DBNs may have different structures, instead of directly comparing the two likelihoods, we used the difference between these two likelihoods for classification. More specifically, during training, for each training sequence $TR_i$, we computed the ratio of two likelihoods $R_i^{TR} = P_c^i/P_s^i$ where $P_c^i = P(TR_i|B_c)$ and $P_s^i = P(TR_i|B_s)$. As expected, generally the ratios of Group C training data were significantly greater than those of Group S. During testing, the ratio $R_{test} = P_c^{test}/P_s^{test}$ for each test sequence was also computed and compared to the ratios of the training data for classification. Fig. 4 reports the classification rates of the different DBNs on each data set. For comparison, the k-th Nearest Neighbor (KNN) algorithm was applied on the fMRI sequences directly and Fig. 4 shows that by using DBNs, classification rates are significantly better with DML-HMM outperforming all other models.

## 5. Conclusions and Future Work

In this work, we contributed a framework of exploiting Dynamic Bayesian Networks to model the functional information of the fMRI data. We explored four types of DBNs: a Parallel Hidden Markov Model (PaHMM), a Coupled Hidden Markov Model (CHMM),

a Fully-linked Hidden Markov Model (FHMM) and a Dynamically Multi-linked Hidden Markov Model. Furthermore, we proposed and compared two structural learning schemes of DML-HMMs and applied the DBNs to a group classification problem. To our knowledge, this is the first time that Dynamic Bayesian Networks are exploited in modeling the connectivity and interactivity among brain voxels from fMRI data. This framework of exploring functional information of fMRI data provides a novel approach of revealing brain connectivity and interactivity and provides an independent test for the effect of psychopathology on brain function.

Currently, DBNs use independently pre-selected brain regions, thus some other important interactivity information may have been discarded in the feature selection step. Our future work will focus on developing a dynamic neuronal circuit modeling framework performing feature selection and DBN learning simultaneously. Due to computational limits and for clarity purposes, we explored only 5 brain regions and thus another direction of future work is to develop a hierarchical DBN topology to comprehensively model all implicated brain regions efficiently.

## References

[1] S. Anders, M. Lotze, M. Erb, W. Grodd, and N. Birbaumer. Brain activity underlying emotional valence and arousal: A response-related fmri study. In *Human Brain Mapping*.

[2] M. Brand, N. Oliver, and A. Pentland. Coupled hidden markov models for complex action recognition. In *CVPR*, pages 994–999, 1996.

[3] J. Ford, H. Farid, F. Makedon, L.A. Flashman, T.W. McAllister, V. Megalooikonomou, and A.J. Saykin. Patient classification of fmri activation maps. In *MICCAI*, 2003.

[4] N. Friedman. The bayesian structual algorithm. In *UAI*, 1998.

[5] N. Friedman, K. Murphy, and S. Russell. Learning the structure of dynamic probabilistic networks. In *Uncertainty in AI*, pages 139–147, 1998.

[6] K. Friston, A. Holmes, K. Worsley, and et al. Statistical parametric maps in functional imaging: A general linear approach. *Human Brain Mapping*, pages 2:189–210, 1995.

[7] G. Ghahramani. Learning dynamic bayesian networks. In *Adaptive Processing of Sequences and Data Structures, Lecture Notes in AI*, pages 168–197, 1998.

[8] R.Z. Goldstein and N.D. Volkow. Drug addiction and its underlying neurobiological basis: Neuroimaging evidence for the involvement of the frontal cortex. *American Journal of Psychiatry*, (10):1642–1652.

[9] R.Z. Goldstein et al. A modified role for the orbitofrontal cortex in attribution of salience to monetary reward in cocaine addiction: an fmri study at 4t. In *Human Brain Mapping Conference*, 2004.

[10] S. Gong and T. Xiang. Recognition of group activities using dynamic probabilistic networks. In *ICCV*, 2003.

[11] M.I. Jordan and Y. Weiss. *Graphical models: probabilistic inference, Arbib, M. (ed): Handbook of Neural Networks and Brain Theory*. MIT Press, 2002.

[12] A.W. MacDonald et al. Dissociating the role of the dorsolateral prefrontal and anterior cingulate cortex in cognitive control. *Science*, 288(5472):1835–1838, 2000.

[13] T.M. Mitchell, R. Hutchinson, R. Niculescu, F. Pereira, X. Wang, M. Just, and S. Newman. Learning to decode cognitive states from brain images. *Machine Learning*, 57:145–175, 2004.

[14] K.P. Murphy. An introduction to graphical models. 2001.

[15] L.K. Hansen P. Hojen-Sorensen and C.E. Rasmussen. Bayesian modeling of fmri time series. In *NIPS*, 1999.

[16] W.D. Penny, K.E. Stephan, A. Mechelli, and K.J. Friston. Comparing dynamic causal models. *NeuroImage*, 22(3):1157–1172, 2004.

[17] C. Vogler and D. Metaxas. A framework for recognizing the simultaneous aspects of american sign language. In *CVIU*, pages 81:358–384, 2001.

[18] X. Wang, R. Hutchinson, and T.M. Mitchell. Training fmri classifiers to detect cognitive states across multiple human subjects. In *NIPS03*, Dec 2003.

[19] L. Zhang, D. Samaras, D. Tomasi, N. Volkow, and R. Goldstein. Machine learning for clinical diagnosis from functional magnetic resonance imaging. In *CVPR*, 2005.
